# Convergent Bounds on the Euclidean Distance

**Yoonho Hwang**        **Hee-Kap Ahn**
Department of Computer Science and Engineering
Pohang University of Science and Technology
POSTECH, Pohang, Gyungbuk, Korea(ROK)
{cypher,heekap}@postech.ac.kr

## Abstract

Given a set $V$ of $n$ vectors in $d$-dimensional space, we provide an efficient method for computing quality upper and lower bounds of the Euclidean distances between a pair of vectors in $V$. For this purpose, we define a distance measure, called the *MS-distance*, by using the mean and the standard deviation values of vectors in $V$. Once we compute the mean and the standard deviation values of vectors in $V$ in $O(dn)$ time, the MS-distance provides upper and lower bounds of Euclidean distance between any pair of vectors in $V$ in constant time. Furthermore, these bounds can be refined further in such a way to converge monotonically to the exact Euclidean distance within *d refinement steps*. An analysis on a random sequence of refinement steps shows that the MS-distance provides very tight bounds in only a few refinement steps. The MS-distance can be used to various applications where the Euclidean distance is used to measure the proximity or similarity between objects. We provide experimental results on the nearest and the farthest neighbor searches.

## 1  Introduction

The Euclidean distance between two vectors $\mathbf{x}$ and $\mathbf{y}$ in $d$-dimensional space is a typical distance measure that reflects their proximity in the space. Measuring the Euclidean distance is a fundamental operation in computer science, including the areas of database, computational geometry, computer vision and computer graphics. In machine learning, the Euclidean distance, denoted by $\mathrm{dist}(\mathbf{x}, \mathbf{y})$, or it's variations(for example, $e^{||\mathbf{x}-\mathbf{y}||}$) are widely used to measure data similarity for clustering [1], classification [2] and so on.

A typical problem is as follows. Given two sets $X$ and $Y$ of vectors in $d$-dimensional space, our goal is to find a pair $(\mathbf{x}, \mathbf{y})$, for $\mathbf{x} \in X$ and $\mathbf{y} \in Y$, such that $\mathrm{dist}(\mathbf{x}, \mathbf{y})$ is the optimum (minimum or maximum) over all such pairs. For the nearest or farthest neighbor searches, $X$ is the set consisting of a single query point while $Y$ consists of all candidate data points. If the dimension is low, a brute-force computation would be fast enough. However, data sets in areas such as optimization, computer vision, machine learning or statistics often live in spaces of dimensionality in the order of thousands or millions. In $d$ dimensional space, a single distance computation already takes $O(d)$ time, thus the cost for finding the nearest or farthest neighbor becomes $O(dnm)$ time, where $n$ and $m$ are the cardinalities of $X$ and $Y$, respectively.

Several techniques have been proposed to reduce computation cost for computing distance. Probably PCA (principal component analysis) is the most frequently used technique for this purpose [3], in which we use an orthogonal transformation based on PCA to convert a set of given data so that the dimensionality of the transformed data is reduced. Then it computes distances between pairs of transformed data efficiently. However, this transformation does not preserve the pairwise distances of data in general, therefore there is no guarantee on the computation results.

If we restrict ourselves to the nearest neighbor search, some methods using space partitioning trees such as KD-tree [4], R-tree [5], or their variations have been widely used. However, they become impractical for high dimensions because of their poor performance in constructing data structures for queries. Recently, cover tree [6] has been used for high dimensional nearest neighbor search, but its construction time increases drastically as the dimension increases [7].

Another approach that has attracted some attention is to compute a good bound of the exact Euclidean distance efficiently such that it can be used to filter off some unnecessary computation, for example, the distance computation between two vectors that are far apart from each other in nearest neighbor search. One of such methods is to compute a distance bound using the inner product approximation [8]. This method, however, requires the distribution of the input data to be known in advance, and works only on data in some predetermined distribution. Another method is to compute a distance bound using bitwise operations [9]. But this method works well only on uniformly distributed vectors, and requires $O(2^d)$ bitwise operations in $d$ dimension. A method using an index structure [10] provides an effective filtering method based on the triangle inequality. But this works well only when data are well clustered.

In this paper, we define a distance measure, called the *MS-distance*, by using the mean and the standard deviation values of vectors in $V$. Once we compute the mean and the standard deviation values of vectors in $V$ in $O(dn)$ time, the MS-distance provides tight upper and lower bounds of Euclidean distance between any pair of vectors in $V$ in constant time. Furthermore, these bounds can be refined further in such a way to converge monotonically to the exact Euclidean distance within $d$ refinement steps. Each refinement step takes constant time.

We provide an analysis on a random sequence of $k$ refinement steps for $0 \leq k \leq d$, which shows a good expectation on the lower and upper bounds. This can justify that the MS-distance provides very tight bounds in a few refinement steps of a typical sequence. We also show that the MS-distance can be used in fast filtering. Note that we do not use any assumption on data distribution.

The MS-distance can be used to various applications where the Euclidean distance is a measure for proximity or similarity between objects. Among them, we provide experimental results on the nearest and the farthest neighbor searches.

## 2 An Upper and A Lower Bounds of the Euclidean Distance

For a $d$-dimensional vector $\mathbf{x} = [x_1, x_2, \ldots, x_d]$, we denote its mean by $\mu_{\mathbf{x}} = \frac{1}{d} \sum_{i=1}^{d} x_i$ and its variance by $\sigma_{\mathbf{x}}^2 = \frac{1}{d} \sum_{i=1}^{d} (x_i - \mu_{\mathbf{x}})^2$. For a pair of vectors $\mathbf{x}$ and $\mathbf{y}$, we can reformulate the squared Euclidean distance between $\mathbf{x}$ and $\mathbf{y}$ as follows. Let $\mathbf{a} = [a_1, a_2, \ldots, a_d]$ and $\mathbf{b} = [b_1, b_2, \ldots, b_d]$ such that $a_i = x_i - \mu_{\mathbf{x}}$ and $b_i = y_i - \mu_{\mathbf{y}}$.

$$\text{dist}(\mathbf{x}, \mathbf{y})^2 = \sum_{i=1}^{d} (x_i - y_i)^2$$

$$= \sum_{i=1}^{d} ((\mu_{\mathbf{x}} + a_i) - (\mu_{\mathbf{y}} + b_i))^2$$

$$= \sum_{i=1}^{d} (\mu_{\mathbf{x}}^2 + 2a_i\mu_{\mathbf{x}} + a_i^2 + \mu_{\mathbf{y}}^2 + 2b_i\mu_{\mathbf{y}} + b_i^2 - 2(\mu_{\mathbf{x}}\mu_{\mathbf{y}} + a_i\mu_{\mathbf{y}} + b_i\mu_{\mathbf{x}} + a_ib_i)) \quad (1)$$

$$= \sum_{i=1}^{d} (\mu_{\mathbf{x}}^2 - 2\mu_{\mathbf{x}}\mu_{\mathbf{y}} + \mu_{\mathbf{y}}^2 + a_i^2 + b_i^2 - 2a_ib_i) \quad (2)$$

$$= d\left((\mu_{\mathbf{x}} - \mu_{\mathbf{y}})^2 + (\sigma_{\mathbf{x}} + \sigma_{\mathbf{y}})^2\right) - 2d\sigma_{\mathbf{x}}\sigma_{\mathbf{y}} - 2\sum_{i=1}^{d} a_ib_i \quad (3)$$

$$= d\left((\mu_{\mathbf{x}} - \mu_{\mathbf{y}})^2 + (\sigma_{\mathbf{x}} - \sigma_{\mathbf{y}})^2\right) + 2d\sigma_{\mathbf{x}}\sigma_{\mathbf{y}} - 2\sum_{i=1}^{d} a_ib_i. \quad (4)$$

By the definitions of $a_i$ and $b_i$, we have $\sum_{i=1}^{d} a_i = \sum_{i=1}^{d} b_i = 0$, and $\frac{1}{d}\sum_{i=1}^{d} a_i^2 = \sigma_{\mathbf{x}}^2$. By the first properties, equation (1) is simplified to (2), and by the second property, equations (2) becomes (3) and (4).

Note that equations (3) and (4) are composed of the mean and variance values (their products and squared values, multiplied by $d$) of $\mathbf{x}$ and $\mathbf{y}$, except the last summations. Thus, once we preprocess $V$ of $n$ vectors such that both $\mu_{\mathbf{x}}$ and $\sigma_{\mathbf{x}}$ for all $\mathbf{x} \in V$ are computed in $O(dn)$ time and stored in a table of size $O(n)$, this sum can be computed in constant time for any pair of vectors, regardless of the dimension.

The last summation, $\sum_{i}^{d} a_i b_i$, is the inner product $\langle \mathbf{a}, \mathbf{b} \rangle$, and therefore by applying the Cauchy-Schwarz inequality we get

$$|\langle \mathbf{a}, \mathbf{b} \rangle| = |\sum_{i=1}^{d} a_i b_i| \leq \sqrt{(\sum_{i=1}^{d} a_i^2)(\sum_{i=1}^{d} b_i^2)} = d\sigma_{\mathbf{x}}\sigma_{\mathbf{y}}. \tag{5}$$

This gives us the following upper and lower bounds of the squared Euclidean distance from equations (3) and (4).

**Lemma 1** *For two d-dimensional vectors $\mathbf{x}, \mathbf{y}$, the followings hold.*

$$\mathrm{dist}(\mathbf{x}, \mathbf{y})^2 \geq d\left((\mu_{\mathbf{x}} - \mu_{\mathbf{y}})^2 + (\sigma_{\mathbf{x}} - \sigma_{\mathbf{y}})^2\right) \tag{6}$$

$$\mathrm{dist}(\mathbf{x}, \mathbf{y})^2 \leq d\left((\mu_{\mathbf{x}} - \mu_{\mathbf{y}})^2 + (\sigma_{\mathbf{x}} + \sigma_{\mathbf{y}})^2\right) \tag{7}$$

## 3 The MS-distance

The lower and upper bounds in inequalities (6) and (7) can be computed in constant time once we compute the mean and standard variance values of each vector in $V$ in the preprocessing. However, in some applications these bounds may not be tight enough. In this section, we introduce the *MS-distance* which not only provides lower and upper bounds of the Euclidean distance in constant time, but also could be refined further in such a way to converge to the exact Euclidean distance within $d$ steps.

To do this, we reformulate the last term of equations (3) and (4), that is, the inner product $\langle \mathbf{a}, \mathbf{b} \rangle$. If the norms $||\mathbf{a}|| = \sqrt{\sum_{i=1}^{d} a_i^2}$ or $||\mathbf{b}|| = \sqrt{\sum_{i=1}^{d} b_i^2}$ are zero, then $\sum_{i=1}^{d} a_i b_i = 0$, thus the upper and lower bounds become the same. This implies that we can compute the exact Euclidean distance in constant time. So from now on, we assume that both $||\mathbf{a}||$ and $||\mathbf{b}||$ are non-zero. We reformulate the inner product $\langle \mathbf{a}, \mathbf{b} \rangle$.

$$\sum_{i=1}^{d} a_i b_i = d\sigma_{\mathbf{x}}\sigma_{\mathbf{y}} - d\sigma_{\mathbf{x}}\sigma_{\mathbf{y}} + \sum_{i=1}^{d} a_i b_i$$

$$= d\sigma_{\mathbf{x}}\sigma_{\mathbf{y}} - \frac{\sigma_{\mathbf{x}}\sigma_{\mathbf{y}}}{2}\left(2d - \sum_{i=1}^{d}\frac{2a_i b_i}{\sigma_{\mathbf{x}}\sigma_{\mathbf{y}}}\right)$$

$$= d\sigma_{\mathbf{x}}\sigma_{\mathbf{y}} - \frac{\sigma_{\mathbf{x}}\sigma_{\mathbf{y}}}{2}\left(\sum_{i=1}^{d}\left(\frac{a_i}{\sigma_{\mathbf{x}}}\right)^2 + \sum_{i=1}^{d}\left(\frac{b_i}{\sigma_{\mathbf{y}}}\right)^2 - \sum_{i=1}^{d}\frac{2a_i b_i}{\sigma_{\mathbf{x}}\sigma_{\mathbf{y}}}\right) \tag{8}$$

$$= d\sigma_{\mathbf{x}}\sigma_{\mathbf{y}} - \frac{\sigma_{\mathbf{x}}\sigma_{\mathbf{y}}}{2}\sum_{i=1}^{d}(\frac{b_i}{\sigma_{\mathbf{y}}} - \frac{a_i}{\sigma_{\mathbf{x}}})^2 \tag{9}$$

$$= -d\sigma_{\mathbf{x}}\sigma_{\mathbf{y}} + \frac{\sigma_{\mathbf{x}}\sigma_{\mathbf{y}}}{2}\sum_{i=1}^{d}(\frac{b_i}{\sigma_{\mathbf{y}}} + \frac{a_i}{\sigma_{\mathbf{x}}})^2 \tag{10}$$

Equation (8) is because of $\sum_{i=1}^{d} a_i^2 = d\sigma_{\mathbf{x}}^2$ and $\sum_{i=1}^{d} b_i^2 = d\sigma_{\mathbf{y}}^2$. We can also get equation (10) by switching the roles of the term $-d\sigma_{\mathbf{x}}\sigma_{\mathbf{y}}$ and the term $d\sigma_{\mathbf{x}}\sigma_{\mathbf{y}}$ in the above equations.

**Definition.** Now we define the MS-distance between $\mathbf{x}$ and $\mathbf{y}$ in its lower bound form, denoted by $\mathrm{MSL}(\mathbf{x}, \mathbf{y}, k)$, by replacing the last term of equation (3) with equation (9), and in its upper bound form, denoted by $\mathrm{MSU}(\mathbf{x}, \mathbf{y}, k)$ by replacing the last term of equation (4) with equation (10). The MS-distance makes use of the nonincreasing intermediate values for its upper bound and the nondecreasing intermediate values for its lower bound. We let $a_0 = b_0 = 0$.

$$\mathrm{MSL}(\mathbf{x}, \mathbf{y}, k) = d \left( (\mu_{\mathbf{x}} - \mu_{\mathbf{y}})^2 + (\sigma_{\mathbf{x}} - \sigma_{\mathbf{y}})^2 \right) + \sigma_{\mathbf{x}} \sigma_{\mathbf{y}} \sum_{i=0}^{k} \left( \frac{b_i}{\sigma_{\mathbf{y}}} - \frac{a_i}{\sigma_{\mathbf{x}}} \right)^2 \tag{11}$$

$$\mathrm{MSU}(\mathbf{x}, \mathbf{y}, k) = d \left( (\mu_{\mathbf{x}} - \mu_{\mathbf{y}})^2 + (\sigma_{\mathbf{x}} + \sigma_{\mathbf{y}})^2 \right) - \sigma_{\mathbf{x}} \sigma_{\mathbf{y}} \sum_{i=0}^{k} \left( \frac{b_i}{\sigma_{\mathbf{y}}} + \frac{a_i}{\sigma_{\mathbf{x}}} \right)^2 \tag{12}$$

**Properties.** Note that equation (11) is nondecreasing and equation (12) is nonincreasing while $i$ increases from 0 to $d$, because $d$, $\sigma_{\mathbf{x}}$, and $\sigma_{\mathbf{y}}$ are all nonnegative, and $(\frac{b_i}{\sigma_{\mathbf{y}}} - \frac{a_i}{\sigma_{\mathbf{x}}})^2$ and $(\frac{b_i}{\sigma_{\mathbf{y}}} + \frac{a_i}{\sigma_{\mathbf{x}}})^2$ are also nonnegative for all $i$. This is very useful because, in equation (11), the first term, $\mathrm{MSL}(\mathbf{x}, \mathbf{y}, 0)$, is already a lower bound of $\mathrm{dist}(\mathbf{x}, \mathbf{y})^2$ by inequality (6) , and the lower bound can be refined further nondecreasingly over the summation in the second term. If we stop the summation at $i = k$, for $k < d$, the intermediate result is also a refined lower bounds of $\mathrm{dist}(\mathbf{x}, \mathbf{y})^2$. Similarly, in equation (12), the first term, $\mathrm{MSU}(\mathbf{x}, \mathbf{y}, 0)$, is already an upper bound of $\mathrm{dist}(\mathbf{x}, \mathbf{y})^2$ by inequality (7) , and the upper bound can be refined further nonincreasingly over the summation in the second term. This means we can stop the summation as soon as we find a bound good enough for the application under consideration. If we need the exact Euclidean distance, we can get it by continuing to the full summation. We summarize the above properties in the following.

**Lemma 2** *(Monotone Convergence) Let* $\mathrm{MSL}(\mathbf{x}, \mathbf{y}, k)$ *and* $\mathrm{MSU}(\mathbf{x}, \mathbf{y}, k)$ *be the lower and upper bounds of MS-distance as defined above, respectively. Then the following properties hold.*

- $\mathrm{MSL}(\mathbf{x}, \mathbf{y}, 0) \leq \mathrm{MSL}(\mathbf{x}, \mathbf{y}, 1) \leq \cdots \leq \mathrm{MSL}(\mathbf{x}, \mathbf{y}, d-1) \leq \mathrm{MSL}(\mathbf{x}, \mathbf{y}, d) = \mathrm{dist}(\mathbf{x}, \mathbf{y})^2$.

- $\mathrm{MSU}(\mathbf{x}, \mathbf{y}, 0) \geq \mathrm{MSU}(\mathbf{x}, \mathbf{y}, 1) \geq \cdots \geq \mathrm{MSU}(\mathbf{x}, \mathbf{y}, d-1) \geq \mathrm{MSU}(\mathbf{x}, \mathbf{y}, d) = \mathrm{dist}(\mathbf{x}, \mathbf{y})^2$.

- $\mathrm{MSL}(\mathbf{x}, \mathbf{y}, k) = \mathrm{MSL}(\mathbf{x}, \mathbf{y}, k+1)$ *if and only if* $b_{k+1}/\sigma_{\mathbf{y}} = a_{k+1}/\sigma_{\mathbf{x}}$.

- $\mathrm{MSU}(\mathbf{x}, \mathbf{y}, k) = \mathrm{MSU}(\mathbf{x}, \mathbf{y}, k+1)$ *if and only if* $b_{k+1}/\sigma_{\mathbf{y}} = -a_{k+1}/\sigma_{\mathbf{x}}$.

**Lemma 3** *For* $0 \leq k < d$*, we can update* $\mathrm{MSL}(\mathbf{x}, \mathbf{y}, k)$ *to* $\mathrm{MSL}(\mathbf{x}, \mathbf{y}, k+1)$*, and* $\mathrm{MSU}(\mathbf{x}, \mathbf{y}, k)$ *to* $\mathrm{MSU}(\mathbf{x}, \mathbf{y}, k+1)$ *in constant time.*

**Fast Filtering.** We must emphasize that $\mathrm{MSL}(\mathbf{x}, \mathbf{y}, 0)$ and $\mathrm{MSU}(\mathbf{x}, \mathbf{y}, 0)$ can be used for fast filtering. Let $\phi$ denote a threshold for filtering defined in some proximity search problem under consideration. If $\phi < \mathrm{MSL}(\mathbf{x}, \mathbf{y}, 0)$ in case of nearest search or $\phi > \mathrm{MSL}(\mathbf{x}, \mathbf{y}, 0)$ in case of farthest search, we do not need to consider this pair $(\mathbf{x}, \mathbf{y})$ as a candidate, thus we can save time from computing their exact Euclidean distance.

Precisely speaking, we map each $d$-dimensional vector $\mathbf{x} = [x_1, x_2, \ldots, x_d]$ into a pair of points, $\hat{\mathbf{x}}^+$ and $\hat{\mathbf{x}}^-$, in the 2-dimensional plane such that $\hat{\mathbf{x}}^+ = [\mu_{\mathbf{x}}, \sigma_{\mathbf{x}}]$ and $\hat{\mathbf{x}}^- = [\mu_{\mathbf{x}}, -\sigma_{\mathbf{x}}]$. Then

$$\mathrm{dist}(\hat{\mathbf{x}}^+, \hat{\mathbf{y}}^+)^2 = \mathrm{MSL}(\mathbf{x}, \mathbf{y}, 0)/d \tag{13}$$

$$\mathrm{dist}(\hat{\mathbf{x}}^+, \hat{\mathbf{y}}^-)^2 = \mathrm{MSU}(\mathbf{x}, \mathbf{y}, 0)/d. \tag{14}$$

To see why it is useful in fast filtering, consider the case of finding the nearest vector. For $d$-dimensional vectors in $V$ of size $n$, we have $n$ pairs of points in the plane as in Figure 1. Since $\sigma_{\mathbf{x}}$ is nonnegative, exactly $n$ points lie on or below $\mu$-axis. Let $\mathbf{q}$ be a query vector, and let $\hat{\mathbf{q}}^+$ denote the point mapped in the plane as defined above. Among these $n$ points lying on or below $\mu$-axis, let $\hat{\mathbf{x}}_i^-$ be the point that is nearest to $\hat{\mathbf{q}}^+$. Note that the closest point from the query can be computed efficiently in 2-dimensional space, for example, after constructing some space partitioning structures such as kd-trees or R-trees, each query can be answered in poly-logarithmic search time.

Then we can ignore all $d$-dimensional vectors $\mathbf{x}$ whose mapped point $\hat{\mathbf{x}}^+$ lies outside the circle centered at $\hat{\mathbf{q}}^+$ and of radius $\text{dist}(\hat{\mathbf{q}}^+, \hat{\mathbf{x}}_i^-)$ in the plane, because they are strictly farther than $\mathbf{x}_i$ from $\mathbf{q}$.

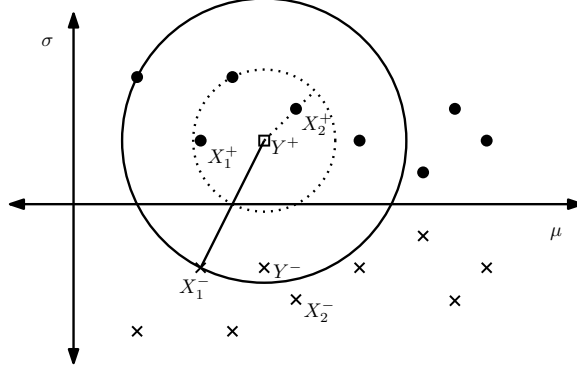

Figure 1: Fast filtering using $\text{MSL}(\mathbf{x}, \mathbf{y}, 0)$ and $\text{MSU}(\mathbf{x}, \mathbf{y}, 0)$. All $d$-dimensional vectors $\mathbf{x}$ whose mapped point $\hat{\mathbf{x}}^+$ lies outside the circle are strictly farther than $\mathbf{x}_i$ from $\mathbf{q}$.

## 4 Estimating the Expected Difference Between Two Bounds

We now turn to estimating the expected difference between $\text{MSL}(\mathbf{x}, \mathbf{y}, k)$ and $\text{MSU}(\mathbf{x}, \mathbf{y}, k)$. Observe that $\text{MSL}(\mathbf{x}, \mathbf{y}, k)$ is almost the same as $\text{MSL}(\mathbf{x}, \mathbf{y}, k-1)$ if $b_k/\sigma_{\mathbf{y}} \approx a_k/\sigma_{\mathbf{x}}$. Hence, in the worst case, $\text{MSL}(\mathbf{x}, \mathbf{y}, 0) = \text{MSL}(\mathbf{x}, \mathbf{y}, d-1) < \text{MSL}(\mathbf{x}, \mathbf{y}, d) = \text{dist}(\mathbf{x}, \mathbf{y})^2$ when $b_k/\sigma_{\mathbf{y}} = a_k/\sigma_{\mathbf{x}}$ for all $k = 0, 1, \ldots, d-1$, except $k = d$. Therefore, if we need a lower bound strictly better than $\text{MSL}(\mathbf{x}, \mathbf{y}, 0)$, then we need to go through all $d$ refinement steps, which takes $O(d)$ time. It is not difficult to see that this also applies to the case of $\text{MSU}(\mathbf{x}, \mathbf{y}, k)$.

However, this is unlikely to happen. Consider a random order for the last term in equation $\text{MSL}(\mathbf{x}, \mathbf{y}, k)$ and for the last term in equation $\text{MSU}(\mathbf{x}, \mathbf{y}, k)$. We show below that their expected values increase and decrease linearly, respectively, as $k$ increases from 0 to $d$. Formally, let $(a_{\gamma(i)}, b_{\gamma(i)})$ denote the $i$th pair in the random order. We measure the expected quality of the bounds by the difference between the bounds, that is, $\text{MSU}(\mathbf{x}, \mathbf{y}, k) - \text{MSL}(\mathbf{x}, \mathbf{y}, k)$ as follows.

$$\text{MSU}(\mathbf{x}, \mathbf{y}, k) - \text{MSL}(\mathbf{x}, \mathbf{y}, k) = 4d\sigma_{\mathbf{x}}\sigma_{\mathbf{y}} - 2\sigma_{\mathbf{x}}\sigma_{\mathbf{y}} \sum_{i=0}^{k} \left( \left(\frac{a_{\gamma(i)}}{\sigma_{\mathbf{x}}}\right)^2 + \left(\frac{b_{\gamma(i)}}{\sigma_{\mathbf{y}}}\right)^2 \right) \quad (15)$$

$$= 4d\sigma_{\mathbf{x}}\sigma_{\mathbf{y}} - 2\sigma_{\mathbf{x}}\sigma_{\mathbf{y}}\frac{k}{d} \sum_{i=0}^{d} \left( \left(\frac{a_i}{\sigma_{\mathbf{x}}}\right)^2 + \left(\frac{b_i}{\sigma_{\mathbf{y}}}\right)^2 \right) \quad (16)$$

$$= 4d\sigma_{\mathbf{x}}\sigma_{\mathbf{y}} - 4k\sigma_{\mathbf{x}}\sigma_{\mathbf{y}} \quad (17)$$

$$= 4\sigma_{\mathbf{x}}\sigma_{\mathbf{y}}(d-k) \quad (18)$$

Let us explain how we get Equation (16) from (15). Let $N$ denote the set of all pairs, and let $N^k$ denote the set of first $k$ pairs in the random order. Since each pair in $N$ is treated equally, $N^k$ is a random subset of $N$ of size $k$. Therefore, $\sum_{i=1}^{k} (a_{\gamma(i)}/\sigma_{\mathbf{x}})^2$ is equivalent to take the total sum of $(a_i/\sigma_{\mathbf{x}})^2$ with $i$ from 1 to $d$ and divide it by $d/k$. We can also show this for $\sum_{i=1}^{k} (b_{\gamma(i)}/\sigma_{\mathbf{y}})^2$ by a similar augment.

Equations (17) and (18) are because $\sum_{i=1}^{d} a_i^2 = d\sigma_{\mathbf{x}}^2$ and $\sum_{i=1}^{d} b_i^2 = d\sigma_{\mathbf{y}}^2$ by definitions of $a_i$ and $b_i$. By replacing each squared sum with $d$, that is , by applying $\sum_{i=1}^{d} (a_i/\sigma_{\mathbf{x}})^2 = \sum_{i=1}^{d} (b_i/\sigma_{\mathbf{y}})^2 = d$, we have Equation (18).

**Lemma 4** *The expected value of* $\text{MSU}(\mathbf{x}, \mathbf{y}, k) - \text{MSL}(\mathbf{x}, \mathbf{y}, k)$ *is* $4\sigma_{\mathbf{x}}\sigma_{\mathbf{y}}(d-k)$.

Because $\text{dist}(\mathbf{x}, \mathbf{y})^2$ always lies in between the two bounds, the following also holds.

**Corollary 1** *Both expected values of* $\text{MSU}(\mathbf{x}, \mathbf{y}, k) - \text{dist}(\mathbf{x}, \mathbf{y})^2$ *and* $\text{dist}(\mathbf{x}, \mathbf{y})^2 - \text{MSL}(\mathbf{x}, \mathbf{y}, k)$ *are at most* $4\sigma_{\mathbf{x}}\sigma_{\mathbf{y}}(d - k)$.

This shows a good theoretical expectation on the lower and upper bounds. This can justify that the MS-distance provides very tight bounds in a few refinement steps of a typical sequence.

## 5 Applications : Proximity Searches

The MS-distance can be used to application problems where the Euclidean distance is a measure for proximity or similarity of objects. As a case study, we implemented the nearest neighbor search (NNS) and the farthest neighbor search (FNS) using the MS-distance.

Given a set $X$ of $d$-dimensional vectors $\mathbf{x}_i$, for $i = 1, \ldots, n$, and a $d$-dimensional query vector $\mathbf{q}$, we use the following simple randomized algorithm for NNS. Initially, we set $\phi$ to the threshold given from the application under consideration or computed from the fast filtering in 2-dimension in Section 3.

1. Consider the vectors in $X$ one at a time according to this sequence. At the $i$th stage, we do the followings.

> **if** $\text{MSL}(\mathbf{q}, \mathbf{x}_i, 0) < \phi$ :
>     **for** $j = 1, 2, ..., d$ :
>         **if** $\text{MSL}(\mathbf{q}, \mathbf{x}_i, j) > \phi$ :
>             **break;**
>     **if** $j = d$:
>         $\phi = \text{MSL}(\mathbf{q}, \mathbf{x}_i, d)$;
>         NN $= i$;

2. return NN as the nearest neighbor of $\mathbf{q}$ with the squared Euclidean distance $\phi$.

Note that the first line of the pseudocodes filters out the vectors whose distance to $\mathbf{q}$ is larger than $\phi$ as in the fast filtering in Section 3. In the **for** loop, we compute $\text{MSL}(\mathbf{q}, \mathbf{x}_i, j)$ from $\text{MSL}(\mathbf{q}, \mathbf{x}_i, j-1)$ in constant time. From the last two lines of the pseudocodes, we update $\phi$ to the exact Euclidean distance between $\mathbf{q}$ and $\mathbf{x}_i$ and store the index as the *current* nearest neighbor (NN). The algorithm for the farthest neighbor search is similar to this one, except that it uses $\text{MSU}(\mathbf{x}_i, \mathbf{y}, j)$ and maintains the maximum distance.

For empirical comparison, we implemented a linear search algorithm that simply computes distances from $\mathbf{q}$ to every $\mathbf{x}_i$ and chooses the one with the minimum distance. We also used the implementation of the cover tree [6]. A cover tree is a data structure that supports fast nearest neighbor queries given a fixed intrinsic dimensionality [7].

We tested these implementations on data sets from UCI machine learning archive [11]. We selected data sets $D$ from various dimensions (from 10 to $100,000$), and randomly selected 30 queries points $Q \subset D$, and queried them on $D \setminus Q$. We labelled the data set on $d$-dimension as "D$d$". The data sets D500, D5000, D10000, D20000, D100000 were used in *NIPS 2003 challenge on feature selection* [12]. The test machine has one CPU, Intel Q6600 with 2.4GHz, 3GB memory, and 32bit Ubuntu 10 operating system running on the machine.

Figure 2 shows the percentage of data filtered off. For the data sets on relaxed dimensions, the MS-distance filtered off over $95\%$ of data without lose of accuracy. For high dimensional data, MS-distance failed to filter off many data. Probably this is because the distances from queries to their nearest vectors tend to converge to the distances to their farthest vectors as described in [13]. This makes it hard to decrease (or increase in FNS) the threshold $\phi$ for the MS-distance enough to filter off many data. However, on such high dimensions, both the linear search and the cover tree algorithm also show poor performance.

Figure 3 shows the preprocessing time of the MS-distance and the cover tree for NNS. The time axis is log-scaled second. This shows that the preprocessing time of the MS-distance is up to 1000 times

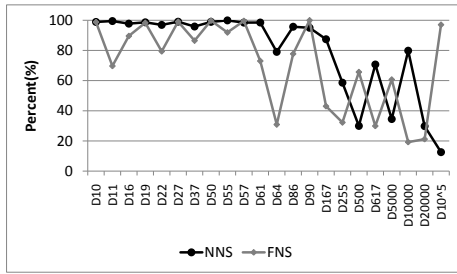

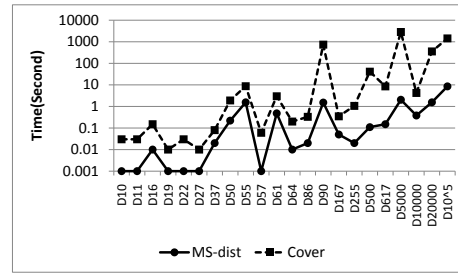

Figure 2: Data filtered off in percentage.

Figure 3: Preprocessing time for nearest neighbor search in log-scaled second.

faster than the one in the cover tree. This is because for the MS-distance it requires only $O(dn)$ time to compute the mean and the standard deviation values.

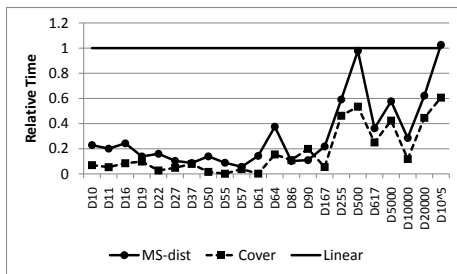

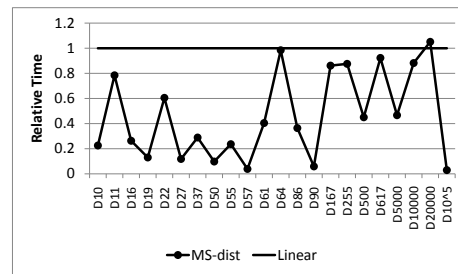

Figure 4: Relative running time for the nearest neighbor search queries, normalized by linear search time.

Figure 5: Relative running time for the farthest neighbor search queries, normalized by linear search time.

Figure 4 shows the time spent for NNS queries. The graph shows the query time that is normalized by the linear search time. It is clear that the filtering algorithm based on the MS-distance beats the linear search algorithm, even on high dimensional data in the results. The cover tree, which is designed exclusively for NNS, shows slightly better query performance than ours. However, the MS-distance is more general and flexible: it supports addition of a new vector to the data set (our data structure) in $O(d)$ time for computing the mean and the standard deviation values of the vector. Deletion of a vector from the data set can be done in constant time. Furthermore, the data structure for NNS can also be used for FNS.

Figure 5 shows the time spent for FNS queries. This is outstanding compared to the linear search algorithm. We hardly know any other previous work achieving better performance than this.

## 6 Conclusion

We introduce a fast distance bounding technique, called the MS-distance, by using the mean and the standard deviation values. The MS-distance between two vectors provides upper and lower bounds of Euclidean distance between them in constant time, and these bounds converge monotonically to the exact Euclidean distance over iteration. The MS-distance can be used to application problems where the Euclidean distance is a measure for proximity or similarity of objects. The experimental results show that our method is efficient enough even to replace the best known algorithms for proximity searches.

Table 1: Data sets

| Data Label | Name | # of vectors | Data Label | Name | # of vectors |
|---|---|---|---|---|---|
| D10 | Page Blocks | 5473 | D64 | Optical Recognition | 5620 |
| D11 | Wine Quality | 6497 | D86 | Insurance Company | 5822 |
| D16 | Letter Recognition | 20000 | D90 | YearPredictionMSD | 515345 |
| D19 | Image Segmentation | 2310 | D167 | Musk2 | 6597 |
| D22 | Parkinsons Tel | 5875 | D255 | Semeion | 1593 |
| D27 | Steel Plates Faults | 1941 | D500 | Madelon | 4400 |
| D37 | Statlog Satellite | 6435 | D617 | ISOLET | 7795 |
| D50 | MiniBooNE | 130064 | D5000 | Gisette | 13500 |
| D55 | Covertype | 581012 | D10000 | Arcene | 900 |
| D57 | Spambase | 4601 | D20000 | Dexter | 2600 |
| D61 | IPUMS Census | 233584 | D100000 | Dorothea | 1950 |

**Acknowledgments**

This work was supported by the National Research Foundation of Korea Grant funded by the Korean Government (MEST) (NRF-2010-0009857).

# References

[1] J. B. MacQueen. Some methods for classification and analysis of multivariate observations. In L. M. Le Cam and J. Neyman, editors, *Proceedings of the fifth Berkeley Symposium on Mathematical Statistics and Probability*, volume 1, pages 281–297. University of California Press, 1967.

[2] V. N. Vapnik. *The Nature of Statistical Learning Theory*. Springer, New York, NY, USA, 1995.

[3] K. Pearson. On lines and planes of closest fit to systems of points in space. *Philosophical Magazine*, 2:559–572, 1901.

[4] J. L. Bentley. Multidimensional binary search trees used for associative searching. *Communications of ACM*, 18:509–517, September 1975.

[5] A. Guttman. R-trees: A dynamic index structure for spatial searching. In Beatrice Yormark, editor, *Proceedings of the ACM SIGMOD International Conference on Management of Data*, SIGMOD '84, pages 47–57. ACM, 1984.

[6] A. Beygelzimer, S. Kakade, and J. Langford. Cover trees for nearest neighbor. In *Proceedings of the 23rd international conference on Machine learning*, ICML '06, pages 97–104, New York, NY, USA, 2006. ACM.

[7] D. R. Karger and M. Ruhl. Finding nearest neighbors in growth-restricted metrics. In *Proceedings of the 34th annual ACM symposium on Theory of computing*, STOC '02, pages 741–750, New York, NY, USA, 2002. ACM.

[8] Ö. Eğecioğlu and H. Ferhatosmanoğlu. Dimensionality reduction and similarity computation by inner product approximations. In *Proceedings of the ninth international conference on Information and knowledge management*, CIKM '00, pages 219–226, New York, NY, USA, 2000. ACM.

[9] R. Weber, H. J. Schek, and S. Blott. A quantitative analysis and performance study for similarity-search methods in high-dimensional spaces. In *Proceedings of the 24rd International Conference on Very Large Data Bases*, VLDB '98, pages 194–205, San Francisco, CA, USA, 1998. Morgan Kaufmann Publishers Inc.

[10] H. V. Jagadish, B. C. Ooi, K.L. Tan, C. Yu, and R. Zhang. idistance: An adaptive b+-tree based indexing method for nearest neighbor search. *ACM Transactions on Database Systems*, 30:364–397, June 2005.

[11] UCI machine learning archive. http://archive.ics.uci.edu/ml/.

[12] NIPS 2003 challenge on feature selection. http://clopinet.com/isabelle/projects/nips2003/.

[13] K. S. Beyer, J. Goldstein, R. Ramakrishnan, and U. Shaft. When is "nearest neighbor" meaningful? In *Proceedings of the 7th International Conference on Database Theory*, ICDT '99, pages 217–235, London, UK, 1999. Springer.

